# Relational Learning with Gaussian Processes

**Wei Chu**
CCLS
Columbia Univ.
New York, NY 10115

**Vikas Sindhwani**
Dept. of Comp. Sci.
Univ. of Chicago
Chicago, IL 60637

**Zoubin Ghahramani**
Dept. of Engineering
Univ. of Cambridge
Cambridge, UK

**S. Sathiya Keerthi**
Yahoo! Research
Media Studios North
Burbank, CA 91504

## Abstract

Correlation between instances is often modelled via a kernel function using input attributes of the instances. Relational knowledge can further reveal additional pairwise correlations between variables of interest. In this paper, we develop a class of models which incorporates both reciprocal relational information and input attributes using Gaussian process techniques. This approach provides a novel non-parametric Bayesian framework with a data-dependent covariance function for supervised learning tasks. We also apply this framework to semi-supervised learning. Experimental results on several real world data sets verify the usefulness of this algorithm.

## 1 Introduction

Several recent developments such as the growth of the world wide web and the maturation of genomic technologies, have brought new domains of application to machine learning research. Many such domains involve *relational* data in which instances have "links" or inter-relationships between them that are highly informative for learning tasks, e.g. (Taskar et al., 2002). For example, hyper-linked web-documents are often about similar topics, even if their textual contents are disparate when viewed as bags of words. In document categorization, the citations are important as well since two documents referring to the same reference are likely to have similar content. In computational biology, knowledge about physical interactions between proteins can supplement genomic data for developing good similarity measures for protein network inference. In such cases, a learning algorithm can greatly benefit by taking into account the global network organization of such inter-relationships rather than relying on input attributes alone.

One simple but general type of relational information can be effectively represented in the form of a graph $\mathcal{G} = (\mathcal{V}, \mathcal{E})$. The vertex set $\mathcal{V}$ represents a collection of input instances (which may contain the labelled inputs as a subset, but is typically a much larger set of instances). The edge set $\mathcal{E} \subset \mathcal{V} \times \mathcal{V}$ represents the pairwise relations over these input instances. In this paper, we restrict our attention to undirected edges, i.e., reciprocal relations, though directionality may be an important aspect of some relational datasets. These undirected edges provide useful *structural* knowledge about correlation between the vertex instances. In particular, we allow edges to be of two types – "positive" or "negative" depending on whether the associated adjacent vertices are positively or negatively correlated, respectively. On many problems, only positive edges may be available.

This setting is also applicable to semi-supervised tasks even on traditional "flat" datasets where the linkage structure may be derived from data input attributes. In graph-based semi-supervised methods, $\mathcal{G}$ is typically an adjacency graph constructed by linking each instance (including labelled and unlabelled) to its neighbors according to some distance metric in the input space. The graph $\mathcal{G}$ then serves as an estimate of the global geometric structure of the data. Many algorithmic frameworks for semi-supervised (Sindhwani et al., 2005) and transductive learning, see e.g. (Zhou et al., 2004; Zhu et al., 2003), have been derived under the assumption that data points nearby on this graph are positively correlated.

Several methods have been proposed recently to incorporate relational information within learning algorithms, e.g. for clustering (Basu et al., 2004; Wagstaff et al., 2001), metric learning (Bar-Hillel et al., 2003), and graphical modeling (Getoor et al., 2002). The reciprocal relations over input instances essentially reflect the network structure or the distribution underlying the data, which enrich our prior belief of how instances in the entire input space are correlated. In this paper, we integrate relational information with input attributes in a non-parametric Bayesian framework based on Gaussian processes (GP) (Rasmussen & Williams, 2006), which leads to a data-dependent covariance/kernel function. We highlight the following aspects of our approach: **1)** We propose a novel likelihood function for undirected linkages and carry out approximate inference using efficient Expectation Propagation techniques under a Gaussian process prior. The covariance function of the approximate posterior distribution defines a relational Gaussian process, hereafter abbreviated as RGP. RGP provides a novel Bayesian framework with a data-dependent covariance function for supervised learning tasks. We also derive explicit formulae for linkage prediction over pairs of test points. **2)** When applied to semi-supervised learning tasks involving labelled and unlabelled data, RGP is closely related to the warped reproducing kernel Hilbert Space approach of (Sindhwani et al., 2005) using a novel graph regularizer. Unlike many recently proposed graph-based Bayesian approaches, e.g. (Zhu et al., 2003; Krishnapuram et al., 2004; Kapoor et al., 2005), which are mainly transductive by design, RGP delineates a decision boundary in the input space and provides probabilistic induction over unseen test points. Furthermore, by maximizing the joint evidence of known labels and linkages, we explicitly involve unlabelled data in the model selection procedure. Such a semi-supervised hyper-parameter tuning method can be very useful when there are very few, possibly noisy labels. **3)** On a variety of classification tasks, RGP requires very few labels for providing high-quality generalization on unseen test examples as compared to standard GP classification that ignores relational information. We also report experimental results on semi-supervised learning tasks comparing with competitive deterministic methods.

The paper is organized as follows. In section 2 we develop relational Gaussian processes. Semi-supervised learning under this framework is discussed in section 3. Experimental results are presented in section 4. We conclude this paper in section 5.

## 2 Relational Gaussian Processes

In the standard setting of learning from data, instances are usually described by a collection of input attributes, denoted as a column vector $\mathbf{x} \in \mathcal{X} \subset \mathbb{R}^d$. The key idea in Gaussian process models is to introduce a random variable $f_{\mathbf{x}}$ for all points in the input space $\mathcal{X}$. The values of these random variables $\{f_{\mathbf{x}}\}_{\mathbf{x} \in \mathcal{X}}$ are treated as outputs of a zero-mean Gaussian process. The covariance between $f_{\mathbf{x}}$ and $f_{\mathbf{z}}$ is fully determined by the coordinates of the data pair $\mathbf{x}$ and $\mathbf{z}$, and is defined by any Mercer kernel function $\mathcal{K}(\mathbf{x}, \mathbf{z})$. Thus, the prior distribution over $\boldsymbol{f} = [f_{\mathbf{x}_1} \ldots f_{\mathbf{x}_n}]$ associated with any collection of $n$ points $\mathbf{x}_1 \ldots \mathbf{x}_n$ is a multivariate Gaussian, written as

$$\mathcal{P}(\boldsymbol{f}) = \frac{1}{(2\pi)^{n/2} \det(\Sigma)^{1/2}} \exp\left(-\frac{1}{2}\boldsymbol{f}^T \Sigma^{-1} \boldsymbol{f}\right) \tag{1}$$

where $\Sigma$ is the $n \times n$ covariance matrix whose $ij$-th element is $\mathcal{K}(\mathbf{x}_i, \mathbf{x}_j)$. In the following, we consider the scenario with undirected linkages over a set of instances.

### 2.1 Undirected Linkages

Let the vertex set $\mathcal{V}$ in the relational graph be associated with $n$ input instances $\mathbf{x}_1 \ldots \mathbf{x}_n$. Consider a set of observed pairwise undirected linkages on these instances, denoted as $\boldsymbol{\mathcal{E}} = \{\mathcal{E}_{ij}\}$. Each linkage is treated as a Bernoulli random variable, i.e. $\mathcal{E}_{ij} \in \{+1, -1\}$. Here $\mathcal{E}_{ij} = +1$ indicates that the instances $\mathbf{x}_i$ and $\mathbf{x}_j$ are "positively tied" and $\mathcal{E}_{ij} = -1$ indicates the instances are "negatively tied".

We propose a new likelihood function to capture these undirected linkages, which is defined as follows:

$$\mathcal{P}_{\text{ideal}}\left(\mathcal{E}_{ij}|f_{\mathbf{x}_i}, f_{\mathbf{x}_j}\right) = \left\{ \begin{array}{ll} 1 & \text{if } f_{\mathbf{x}_i} f_{\mathbf{x}_j} \mathcal{E}_{ij} > 0 \\ 0 & \qquad \text{otherwise} \end{array} \right. \tag{2}$$

This formulation is for ideal, noise-free cases; it enforces that the variable values corresponding to positive and negative edges have the same and opposite signs respectively. In the presence of

uncertainty in observing $\mathcal{E}_{ij}$, we assume the variable values $f_{\mathbf{x}_i}$ and $f_{\mathbf{x}_j}$ are contaminated with Gaussian noise that allows some tolerance for noisy observations. The Gaussian noise is of zero mean and unknown variance $\sigma^2$.[1] Let $\mathcal{N}(\delta; \mu, \sigma^2)$ denote a Gaussian random variable $\delta$ with mean $\mu$ and variance $\sigma^2$. Then the likelihood function (2) becomes

$$\mathcal{P}\left(\mathcal{E}_{ij} = +1 | f_{\mathbf{x}_i}, f_{\mathbf{x}_j}\right) = \int \int \mathcal{P}_{\text{ideal}}\left(\mathcal{E}_{ij} = +1 | f_{\mathbf{x}_i} + \delta_i, f_{\mathbf{x}_j} + \delta_j\right) \mathcal{N}(\delta_i; 0, \sigma^2)\mathcal{N}(\delta_j; 0, \sigma^2)\, d\delta_i\, d\delta_j$$
$$= \Phi\left(\frac{f_{\mathbf{x}_i}}{\sigma}\right)\Phi\left(\frac{f_{\mathbf{x}_j}}{\sigma}\right) + \left(1 - \Phi\left(\frac{f_{\mathbf{x}_i}}{\sigma}\right)\right)\left(1 - \Phi\left(\frac{f_{\mathbf{x}_j}}{\sigma}\right)\right)$$

(3)

where $\Phi(z) = \int_{-\infty}^{z} \mathcal{N}(\gamma; 0, 1)\, d\gamma$. The integral in (3) evaluates the volume of a joint Gaussian in the first and third quadrants where $f_{\mathbf{x}_i}$ and $f_{\mathbf{x}_j}$ have the same sign. Note that $\mathcal{P}\left(\mathcal{E}_{ij} = -1 | f_{\mathbf{x}_i}, f_{\mathbf{x}_j}\right) = 1 - \mathcal{P}\left(\mathcal{E}_{ij} = +1 | f_{\mathbf{x}_i}, f_{\mathbf{x}_j}\right)$ and $\mathcal{P}\left(\mathcal{E}_{ij} = +1 | f_{\mathbf{x}_i}, f_{\mathbf{x}_j}\right) = \mathcal{P}\left(\mathcal{E}_{ij} = +1 | -f_{\mathbf{x}_i}, -f_{\mathbf{x}_j}\right)$.

**Remarks:** One may consider other ways to define a likelihood function for the observed edges. For example, we could define $\mathcal{P}_l(\mathcal{E}_{ij} = +1 | f_{\mathbf{x}_i}, f_{\mathbf{x}_j}) = \frac{1}{1 + \exp(-\nu f_{\mathbf{x}_i} f_{\mathbf{x}_j})}$ where $\nu > 0$. However the computation of the predictive probability (9) and its derivatives becomes complicated with this form. Instead of treating edges as Bernoulli variables, we could consider a graph itself as a random variable and then the probability of observing the graph $\mathcal{G}$ can be simply evaluated as: $\mathcal{P}(\mathcal{G}|\boldsymbol{f}) = \frac{1}{\mathcal{Z}}\exp\left(-\frac{1}{2}\boldsymbol{f}^T \Psi\, \boldsymbol{f}\right)$ where $\Psi$ is a graph-regularization matrix (e.g. graph Laplacian) and $\mathcal{Z}$ is a normalization factor that depends on the variable values $\boldsymbol{f}$. Given that there are numerous graph structures over the instances, the normalization factor $\mathcal{Z}$ is intractable in general cases. In the rest of this paper, we will use the likelihood function developed in (3).

## 2.2 Approximate Inference

Combining the Gaussian process prior (1) with the likelihood function (3), we obtain the posterior distribution as follows,

$$\mathcal{P}(\boldsymbol{f}|\boldsymbol{\mathcal{E}}) = \frac{1}{\mathcal{P}(\boldsymbol{\mathcal{E}})}\mathcal{P}(\boldsymbol{f})\prod_{ij}\mathcal{P}\left(\mathcal{E}_{ij}|f_{\mathbf{x}_i}, f_{\mathbf{x}_j}\right) \tag{4}$$

where $\boldsymbol{f} = [f_{\mathbf{x}_1}, \ldots, f_{\mathbf{x}_n}]^T$ and $ij$ runs over the set of observed undirected linkages. The normalization factor $\mathcal{P}(\boldsymbol{\mathcal{E}}) = \int \mathcal{P}(\boldsymbol{\mathcal{E}}|\boldsymbol{f})\mathcal{P}(\boldsymbol{f})d\boldsymbol{f}$ is known as the evidence of the model parameters that serves as a yardstick for model selection.

The posterior distribution is non-Gaussian and multi-modal with a saddle point at the origin. Clearly the posterior mean is at the origin as well. It is important to note that reciprocal relations update the correlation between examples but never change individual mean. To preserve computational tractability and the true posterior mean, we would rather approximate the posterior distribution as a joint Gaussian centered at the true mean than resort to sampling methods. A family of inference techniques can be applied for the Gaussian approximation. Some popular methods include Laplace approximation, mean-field methods, variational methods and expectation propagation. It is inappropriate to apply the Laplace approximation to this case since the posterior distribution is not unimodal and it is a saddle point at the true posterior mean. The standard mean-field methods are also hard to use due to the pairwise relations in observation. Both the variational methods and the expectation propagation (EP) algorithm (Minka, 2001) can be applied here. In this paper, we employ the EP algorithm to approximate the posterior distribution as *a zero-mean Gaussian*. Importantly this still captures the posterior covariance structure allowing prediction of link presence.

The key idea of our EP algorithm here is to approximate $\mathcal{P}(\boldsymbol{f})\prod_{ij}\mathcal{P}\left(\mathcal{E}_{ij}|f_{\mathbf{x}_i}, f_{\mathbf{x}_j}\right)$ as a parametric product distribution[2] in the form of

$$\mathcal{Q}(\boldsymbol{f}) = \mathcal{P}(\boldsymbol{f})\prod_{ij}\tilde{t}(\boldsymbol{f}_{ij}) = \mathcal{P}(\boldsymbol{f})\prod_{ij}s_{ij}\exp\left(-\frac{1}{2}\boldsymbol{f}_{ij}^T\Pi_{ij}\boldsymbol{f}_{ij}\right)$$

where $ij$ runs over the edge set, $\boldsymbol{f}_{ij} = [f_{\mathbf{x}_i}, f_{\mathbf{x}_j}]^T$, and $\Pi_{ij}$ is a symmetric $2 \times 2$ matrix. The parameters $\{s_{ij}, \Pi_{ij}\}$ in $\{\tilde{t}(\boldsymbol{f}_{ij})\}$ are successively optimized by locally minimizing the Kullback-Leibler divergence,

$$\tilde{t}(\boldsymbol{f}_{ij})^{\text{new}} = \arg\min_{\tilde{t}(\boldsymbol{f}_{ij})} \mathbf{KL}\left(\frac{\mathcal{Q}(\boldsymbol{f})}{\tilde{t}(\boldsymbol{f}_{ij})^{\text{old}}}\mathcal{P}(\mathcal{E}_{ij}|\boldsymbol{f}_{ij}) \,\middle\|\, \frac{\mathcal{Q}(\boldsymbol{f})}{\tilde{t}(\boldsymbol{f}_{ij})^{\text{old}}}\tilde{t}(\boldsymbol{f}_{ij})\right). \tag{5}$$

Since $\mathcal{Q}(\boldsymbol{f})$ is in the exponential family, this minimization can be simply solved by moment matching up to the second order. At the equilibrium the EP algorithm returns a Gaussian approximation to the posterior distribution

$$\mathcal{P}(\boldsymbol{f}|\boldsymbol{\mathcal{E}}) \approx \mathcal{N}(0, \mathcal{A}) \tag{6}$$

where $\mathcal{A} = (\Sigma^{-1} + \Pi)^{-1}$, $\Pi = \sum_{ij} \breve{\Pi}_{ij}$ and $\breve{\Pi}_{ij}$ is an $n \times n$ matrix with four non-zero entries augmented from $\Pi_{ij}$. Note that the matrix $\Pi$ could be very sparse. The normalization factor in this Gaussian approximation serves as approximate model evidence that can be explicitly written as

$$\mathcal{P}(\boldsymbol{\mathcal{E}}) \approx \frac{|\mathcal{A}|^{\frac{1}{2}}}{|\Sigma|^{\frac{1}{2}}} \prod_{ij} s_{ij} \tag{7}$$

The detailed updating formulations have to be omitted here to save space. The approximate evidence (7) holds an upper bound on the true value of $\mathcal{P}(\boldsymbol{\mathcal{E}})$ (Wainwright et al., 2005). Its partial derivatives with respect to the model parameters can be analytically derived (Seeger, 2003) and then a gradient-based procedure can be employed for hyperparameter tuning. Although the EP algorithm is known to work quite well in practice, there is no guarantee of convergence to the equilibrium in general. Opper and Winther (2005) proposed expectation consistent (EC) as a new framework for approximations that requires two tractable distributions matching on a set of moments. We plan to investigate the EC algorithm as future work.

## 2.3 Data-dependent Covariance Function

After approximate inference as outlined above, the posterior process conditioned on $\boldsymbol{\mathcal{E}}$ is explicitly given by a modified covariance function defined in the following proposition.

**Proposition**: Given (6), for any finite collection of data points $\mathcal{X}$, the latent random variables $\{f_{\mathbf{x}}\}_{\mathbf{x} \in \mathcal{X}}$ conditioned on $\boldsymbol{\mathcal{E}}$ have a multivariate normal distribution $\mathcal{N}(0, \tilde{\Sigma})$ where $\tilde{\Sigma}$ is the covariance matrix whose elements are given by evaluating the kernel function $\tilde{\mathcal{K}}(\mathbf{x}, \mathbf{z}) : \mathcal{X} \times \mathcal{X} \mapsto \mathbb{R}$ for $\mathbf{x}, \mathbf{z} \in \mathcal{X}$ given by:

$$\tilde{\mathcal{K}}(\mathbf{x}, \mathbf{z}) = \mathcal{K}(\mathbf{x}, \mathbf{z}) - \boldsymbol{k}_{\mathbf{x}}^T (\mathrm{I} + \Pi\Sigma)^{-1} \Pi \boldsymbol{k}_{\mathbf{z}} \tag{8}$$

where I is an $n \times n$ identity matrix, $\boldsymbol{k}_{\mathbf{x}}$ is the column vector $[\mathcal{K}(\mathbf{x}_1, \mathbf{x}), \dots, \mathcal{K}(\mathbf{x}_n, \mathbf{x})]^T$, $\Sigma$ is an $n \times n$ covariance matrix of the vertex set $\mathcal{V}$ obtained by evaluating the base kernel $\mathcal{K}$, and $\Pi$ is defined as in (6).

A proof of this proposition involves some simple matrix algebra and is omitted for brevity. RGP is obtained by a Bayesian update of a standard GP using relational knowledge, which is closely related to the warped reproducing kernel Hilbert space approach (Sindhwani et al., 2005) using a novel graph regularizer $\Pi$ in place of the standard graph Laplacian. Alternatively, we could simply employ the standard graph Laplacian as an approximation of the matrix $\Pi$. This efficient approach has been studied by (Sindhwani et al., 2007) for semi-supervised classification problems.

## 2.4 Linkage Prediction

Given a RGP, the joint distribution of the random variables $\boldsymbol{f}_{rs} = [f_{\mathbf{x}_r}, f_{\mathbf{x}_s}]^T$, associated with a test pair $\mathbf{x}_r$ and $\mathbf{x}_s$, is a Gaussian as well. The linkage predictive distribution $\mathcal{P}(\boldsymbol{f}_{rs}|\boldsymbol{\mathcal{E}})$ can be explicitly written as a zero-mean bivariate Gaussian $\mathcal{N}(\boldsymbol{f}_{rs}; 0, \tilde{\Sigma}_{rs})$ with covariance matrix

$$\tilde{\Sigma}_{rs} = \left[ \begin{array}{cc} \tilde{\mathcal{K}}(\mathbf{x}_r, \mathbf{x}_r) & \tilde{\mathcal{K}}(\mathbf{x}_r, \mathbf{x}_s) \\ \tilde{\mathcal{K}}(\mathbf{x}_s, \mathbf{x}_r) & \tilde{\mathcal{K}}(\mathbf{x}_s, \mathbf{x}_s) \end{array} \right]$$

where $\tilde{\mathcal{K}}$ is defined as in (8). The predictive probability of having a positive edge can be evaluated as

$$\mathcal{P}(\mathcal{E}_{rs}|\boldsymbol{\mathcal{E}}) = \int \mathcal{P}_{\text{ideal}}(\mathcal{E}_{rs}|\boldsymbol{f}_{rs})\mathcal{N}(\boldsymbol{f}_{rs}; 0, \tilde{\Sigma}_{rs})df_{\mathbf{x}_r}df_{\mathbf{x}_s}$$

which can be simplified as

$$\mathcal{P}(\mathcal{E}_{rs}|\boldsymbol{\mathcal{E}}) = \frac{1}{2} + \frac{\arcsin(\rho\mathcal{E}_{rs})}{\pi} \tag{9}$$

where $\rho = \frac{\tilde{\mathcal{K}}(\mathbf{x}_r, \mathbf{x}_s)}{\sqrt{\tilde{\mathcal{K}}(\mathbf{x}_s, \mathbf{x}_s)\tilde{\mathcal{K}}(\mathbf{x}_r, \mathbf{x}_r)}}$. It essentially evaluates the updated correlation between $f_{\mathbf{x}_r}$ and $f_{\mathbf{x}_s}$ after we learn from the observed linkages.

## 3  Semi-supervised Learning

We now apply the RGP framework for semi-supervised learning where a large collection of unlabelled examples are available and labelled data is scarce. Unlabelled examples often identify data clusters or low-dimensional data manifolds. It is commonly assumed that the labels of points within a cluster or nearby on a manifold are highly correlated (Chapelle et al., 2003; Zhu et al., 2003). To apply RGP, we construct positive reciprocal relations between examples within $K$ nearest neighborhood. $K$ could be heuristically set at the minimal integer of nearest neighborhood that could setup a connected graph over labelled and unlabelled examples, where there is a path between each pair of nodes. Learning on these constructed relational data results in a RGP as described in the previous section (see section 4.1 for an illustration). With the RGP as our new prior, supervised learning can be carried out in a straightforward way. In the following we focus on binary classification, but this procedure is also applicable to regression, multi-class classification and ranking.

Given a set of labelled pairs $\{\mathbf{z}_\ell, y_\ell\}_{\ell=1}^m$ where $y_\ell \in \{+1, -1\}$, the Gaussian process classifier (Rasmussen & Williams, 2006) relates the variable $f_{\mathbf{z}_\ell}$ at $\mathbf{z}_\ell$ to the label $y_\ell$ through a probit noise model, i.e. $\mathcal{P}(y_\ell | f_{\mathbf{z}_\ell}) = \Phi(\frac{y_\ell f_{\mathbf{z}_\ell}}{\sigma_n})$ where $\Phi$ is the cumulative normal and $\sigma_n^2$ specifies the label noise level. Combining the probit likelihood with the RGP prior defined by the covariance function (8), we have the posterior distribution as follows,

$$\mathcal{P}(\boldsymbol{f}_\ell | \mathcal{Y}, \boldsymbol{\mathcal{E}}) = \frac{1}{\mathcal{P}(\mathcal{Y}|\boldsymbol{\mathcal{E}})} \mathcal{P}(\boldsymbol{f}_\ell | \boldsymbol{\mathcal{E}}) \prod_\ell \mathcal{P}(y_\ell | f_{\mathbf{z}_\ell})$$

where $\boldsymbol{f}_\ell = [f_{\mathbf{z}_1}, \ldots, f_{\mathbf{z}_m}]^T$, $\mathcal{P}(\boldsymbol{f}_\ell | \boldsymbol{\mathcal{E}})$ is a zero-mean Gaussian with an $m \times m$ covariance matrix $\tilde{\Sigma}_\ell$ whose entries are defined by (8), and $\mathcal{P}(\mathcal{Y}|\boldsymbol{\mathcal{E}})$ is the normalization factor. The posterior distribution can be approximated as a Gaussian as well, denoted as $\mathcal{N}(\mu, \mathcal{C})$, and the quantity $\mathcal{P}(\mathcal{Y}|\boldsymbol{\mathcal{E}})$ can be evaluated accordingly (Seeger, 2003). The predictive distribution of the variable $f_{\mathbf{z}_t}$ at a test case $\mathbf{z}_t$ then becomes a Gaussian, i.e. $\mathcal{P}(f_{\mathbf{z}_t} | \mathcal{Y}, \boldsymbol{\mathcal{E}}) \approx \mathcal{N}(\mu_t, \sigma_t^2)$, where $\mu_t = \boldsymbol{k}_t \tilde{\Sigma}_\ell^{-1} \mu$ and $\sigma_t^2 = \tilde{\mathcal{K}}(\mathbf{z}_t, \mathbf{z}_t) - \boldsymbol{k}_t^T (\tilde{\Sigma}_\ell^{-1} - \tilde{\Sigma}_\ell^{-1} \mathcal{C} \tilde{\Sigma}_\ell^{-1}) \boldsymbol{k}_t$ with $\boldsymbol{k}_t = [\tilde{\mathcal{K}}(\mathbf{z}_1, \mathbf{z}_t), \ldots, \tilde{\mathcal{K}}(\mathbf{z}_m, \mathbf{z}_t)]^T$. One can compute the Bernoulli distribution over the test label $y_t$ by

$$\mathcal{P}(y_t | \mathcal{Y}, \boldsymbol{\mathcal{E}}) = \Phi\left(\frac{\mu_t}{\sqrt{\sigma_n^2 + \sigma_t^2}}\right). \tag{10}$$

To summarize, we first incorporate linkage information into a standard GP that leads to a RGP, and then perform standard inference with the RGP as the prior in supervised learning. Although we describe RGP in two separate steps, these procedures can be seamlessly merged within the Bayesian framework. As for model selection, it is advantageous to directly use the joint evidence

$$\mathcal{P}(\mathcal{Y}, \boldsymbol{\mathcal{E}}) = \mathcal{P}(\mathcal{Y}|\boldsymbol{\mathcal{E}}) \mathcal{P}(\boldsymbol{\mathcal{E}}), \tag{11}$$

to determine the model parameters (such as the kernel parameter, the edge noise level and the label noise level). Note that $\mathcal{P}(\mathcal{Y}, \boldsymbol{\mathcal{E}})$ explicitly involves unlabelled data for model selection. This can be particularly useful when labelled data is very scarce and possibly noisy.

## 4  Numerical Experiments

We start with a synthetic case to illustrate the proposed algorithm (RGP), and then verify the usefulness of this approach on three real world data sets. Throughout the experiments, we consistently compare with the standard Gaussian process classifier (GPC). RGP and GPC are different in the prior only. We employ the linear kernel $\mathcal{K}(\mathbf{x}, \mathbf{z}) = \mathbf{x} \cdot \mathbf{z}$ or the Gaussian kernel $\mathcal{K}(\mathbf{x}, \mathbf{z}) = \exp\left(-\frac{\kappa}{2}\|\mathbf{x} - \mathbf{z}\|_2^2\right)$, and shift the origin of the kernel space to the empirical mean, i.e. $\mathcal{K}(\mathbf{x}, \mathbf{z}) - \frac{1}{n}\sum_i \mathcal{K}(\mathbf{x}, \mathbf{x}_i) - \frac{1}{n}\sum_i \mathcal{K}(\mathbf{z}, \mathbf{x}_i) + \frac{1}{n^2}\sum_i \sum_j \mathcal{K}(\mathbf{x}_i, \mathbf{x}_j)$ where $n$ is the number of available labelled and unlabelled data. The centralized kernel is then used as base kernel in our experiments. The label noise level $\sigma_n^2$ in the GPC and RGP models is fixed at $10^{-4}$. The edge noise level $\sigma^2$ of the RGP models is usually varied from 5 to 0.05. The optimal setting of the $\sigma^2$ and the $\kappa$ in the Gaussian kernel is determined by the joint evidence (11) in each trial. When constructing undirected $K$ nearest- neighbor graphs, $K$ is fixed at the minimal integer required to have a connected graph.

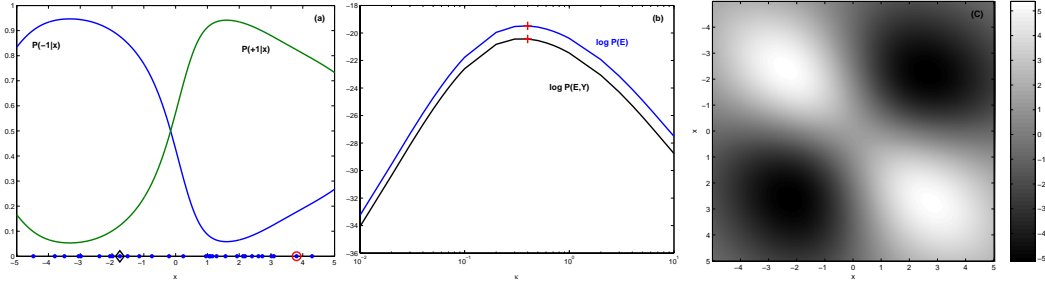

Figure 1: Results on the synthetic dataset. The 30 samples drawn from the Gaussian mixture are presented as dots in (a) and the two labelled samples are indicated by a diamond and a circle respectively. The best $\kappa$ value is marked by the cross in (b). The curves in (a) present the semi-supervised predictive distributions. The prior covariance matrix of RGP learnt from the data is presented in (c).

Table 1: The four universities are Cornell University, the University of Texas at Austin, the University of Washington and the University of Wisconsin. The numbers of categorized Web pages and undirected linkages in the four university dataset are listed in the second column. The averaged AUC scores of label prediction on unlabelled cases are recorded along with standard deviation over 100 trials.

| Task | Web&Link Number | Student or Not | | | Other or Not | | |
|------|-----------------|-----------------|-----------------|-----------------|-----------------|-----------------|-----------------|
| Univ. | Stud Other All Link | GPC | LapSVM | RGP | GPC | LapSVM | RGP |
| Corn. | 128 617 865 13177 | 0.825±0.016 | 0.987±0.008 | 0.989±0.009 | 0.708±0.021 | 0.865±0.038 | 0.884±0.025 |
| Texa. | 148 571 827 16090 | 0.899±0.016 | 0.994±0.007 | 0.999±0.001 | 0.799±0.021 | 0.932±0.026 | 0.906±0.026 |
| Wash. | 126 939 1205 15388 | 0.839±0.018 | 0.957±0.014 | 0.961±0.009 | 0.782±0.023 | 0.828±0.025 | 0.877±0.024 |
| Wisc. | 156 942 1263 21594 | 0.883±0.013 | 0.976±0.029 | 0.992±0.008 | 0.839±0.014 | 0.812±0.030 | 0.899±0.015 |

**4.1 Demonstration**   Suppose samples are distributed as a Gaussian mixture with two components in one-dimensional space, e.g. $0.4 \cdot \mathcal{N}(-2.5, 1) + 0.6 \cdot \mathcal{N}(2.0, 1)$. We randomly collected 30 samples from this distribution, shown as dots on the $x$ axis of Figure 1(a). With $K = 3$, there are 56 "positive" edges over these 30 samples. We fixed $\sigma^2 = 1$ for all the edges, and varied the parameter $\kappa$ from 0.01 to 10. At each setting, we carried out the Gaussian approximation by EP as described in section 2.2. Based on the approximate model evidence $\mathcal{P}(\mathcal{E})$ (7), presented in Figure 1(b), we located the best $\kappa = 0.4$. Figure 1(c) presents the posterior covariance function $\tilde{\mathcal{K}}$ (8) at this optimal setting. Compared to the data-independent prior covariance function defined by the Gaussian kernel, the posterior covariance function captures the density information of the unlabelled samples. The pairs within the same cluster become positively correlated, whereas the pairs between the two clusters turn out to be negatively correlated. This is learnt without any explicit assumption on density distributions. Given two labelled samples, one per class, indicated by the diamond and the circle in Figure 1(a), we carried out supervised learning on the basis of the new prior $\tilde{\mathcal{K}}$, as described in section 3. The joint model evidence $\mathcal{P}(\mathcal{Y}|\mathcal{E})\mathcal{P}(\mathcal{E})$ is plotted out in Figure 1(b). The corresponding predictive distribution (10) with the optimal $\kappa = 0.4$ is presented in Figure 1(a). Note that the decision boundary of the standard GPC should be around $x = 1$. We observed our decision boundary significantly shifts to the low-density region that respects the geometry of the data.

**4.2 The Four University Dataset**   We considered a subset of the WebKB dataset for categorization tasks.[3] The subset, collected from the Web sites of computer science departments of four universities, contains 4160 pages and 9998 hyperlinks interconnecting them. These pages have been manually classified into seven categories: student, course, faculty, staff, department, project and other. The text content of each Web page was preprocessed as bag-of-words, a vector of "term frequency" components scaled by "inverse document frequency", which was used as input attributes. The length of each document vector was normalized to unity. The hyperlinks were translated into 66249 undirected "positive" linkages over the pages under the assumption that two pages are likely to be positively correlated if they are hyper-linked by the same hub page. Note there are no "negative" linkages in this case. We considered two classification tasks, student vs. non-student and other vs. non-other, for each of the four universities. The numbers of samples and linkages of the four universities are listed in Table 1. We randomly selected 10% samples as labelled data and used the remaining samples as unlabelled data. The selection was repeated 100 times. The linear kernel

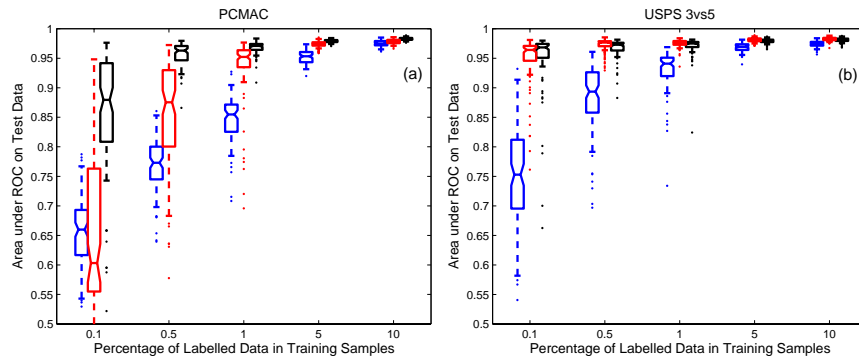

Figure 2: Test AUC results of the two semi-supervised learning tasks, PCMAC in (a) and USPS in (b). The grouped boxes from left to right represent the results of GPC, LapSVM, and RGP respectively at different percentages of labelled samples over 100 trials. The notched-boxes have lines at the lower quartile, median, and upper quartile values. The whiskers are lines extending from each end of the box to the most extreme data value within 1.5 interquartile range. Outliers are data with values beyond the ends of the whiskers, which are displayed as dots.

was used as base kernel in these experiments. We conducted this experiment in a transductive setting where the entire linkage data was used to learn the RGP model and comparisons were made with GPC for predicting labels of unlabelled samples. We make comparisons with a discriminant kernel approach to semi-supervised learning – the Laplacian SVM (Sindhwani et al., 2005) using the linear kernel and a graph Laplacian based regularizer. We recorded the average AUC for predicting labels of unlabelled cases in Table 1.[4] Our RGP models significantly outperform the GPC models by incorporating the linkage information in modelling. RGP is very competitive with LapSVM on "Student or Not" while yields better results on 3 out of 4 tasks of "Other or Not". As future work, it would be interesting to utilize weighted linkages and to compare with other graph kernels.

**4.3 Semi-supervised Learning**   We chose a binary classification problem in the 20 newsgroup dataset, 985 PC documents vs. 961 MAC documents. The documents were preprocessed, same as we did in the previous section, into vectors with 7510 elements. We randomly selected 1460 documents as training data, and tested on the remaining 486 documents. We varied the percentage of labelled data from $0.1\%$ to $10\%$ gradually, and at each percentage repeated the random selection of labelled data 100 times. We used the linear kernel in the RGP and GPC models. With $K = 4$, we got 4685 edges over the 1460 training samples. The test results on the 486 documents are presented in Figure 2(a) as a boxplot. Model parameters for LapSVM were tuned using cross-validation with 50 labelled samples, since it is difficult for discriminant kernel approaches to carry out cross validation when the labelled samples are scarce. Our algorithm yields much better results than GPC and LapSVM, especially when the fraction of labelled data is less than $5\%$. When the labelled samples are few (a typical case in semi-supervised learning), cross validation becomes hard to use while our approach provides a Bayesian model selection by the model evidence.

U.S. Postal Service dataset (USPS) of handwritten digits consists of $16 \times 16$ gray scale images. We focused on constructing a classifier to distinguish digit 3 from digit 5. We used the training/test split, generated and used by (Lawrence & Jordan, 2005), in our experiment for comparison purpose. This partition contains 1214 training samples (556 samples of digit 3 and 658 samples of digit 5) and 326 test samples. With $K = 3$, we obtained 2769 edges over the 1214 training samples. We randomly picked up a subset of the training samples as labelled data and treated the remaining samples as unlabelled. We varied the percentage of labelled data from $0.1\%$ to $10\%$ gradually, and at each percentage repeated the selection of labelled data 100 times. In this experiment, we employed the Gaussian kernel, varied the edge noise level $\sigma^2$ from 5 to 0.5, and tried the following values for $\kappa$, $[0.001, 0.0025, 0.005, 0.0075, 0.01, 0.025, 0.05, 0.075, 0.1]$. The optimal values of $\kappa$ and $\sigma^2$ were decided by the joint evidence $\mathcal{P}(\mathcal{Y}, \mathcal{E})$ (11). We report the error rate and AUC on the 326 test data in Figure 2(b) as a boxplot, along with the test results of GPC and LapSVM. When the percentage of labelled data is less than $5\%$, our algorithm achieved greatly better performance than GPC, and very competitive results compared with LapSVM (tuned with 50 labelled samples) though RGP used

fewer labelled samples in model selection. Comparing with the performance of transductive SVM (TSVM) and the null category noise model for binary classification (NCNM) reported in (Lawrence & Jordan, 2005), we are encouraged to see that our approach outperforms TSVM and NCNM on this experiment.

# 5   Conclusion

We developed a Bayesian framework to learn from relational data based on Gaussian processes. The resulting relational Gaussian processes provide a unified data-dependent covariance function for many learning tasks. We applied this framework to semi-supervised learning and validated this approach on several real world data. While this paper has focused on modelling symmetric (undirected) relations, this relational Gaussian process framework can be generalized for asymmetric (directed) relations as well as multiple classes of relations. Recently, Yu et al. (2006) have represented each relational pair by a tensor product of the attributes of the associated nodes, and have further proposed efficient algorithms. This is a promising direction.

### Acknowledgements

W. Chu is partly supported by a research contract from Consolidated Edison. We thank Dengyong Zhou for sharing the preprocessed Web-KB data.

## Footnotes

[1] We could specify different noise levels for weighted edges. In this paper, we focus on unweighted edges only.

[2] The likelihood function we defined could also be approximated by a Gaussian mixture of two symmetric components, but the difficulty lies in the number of components growing exponentially after multiplication.

[3]The dataset comes from the Web→KB project, see http://www-2.cs.cmu.edu/~webkb/.

[4]AUC stands for the area under the Receiver-Operator Characteristic (ROC) curve.

# References

Bar-Hillel, A., Hertz, T., Shental, N., & Weinshall, D. (2003). Learning distance functions using equivalence relations. *Proceedings of International Conference on Machine Learning* (pp. 11–18).

Basu, S., Bilenko, M., & Mooney, R. J. (2004). A probabilisitic framework for semi-supervised clustering. *Proceedings of ACM SIGKDD International Conference on Knowledge Discovery and Data Mining* (pp. 59–68).

Chapelle, O., Weston, J., & Schölkopf, B. (2003). Cluster kernels for semi-supervised learning. *Neural Information Processing Systems 15* (pp. 585–592).

Getoor, L., Friedman, N., Koller, D., & Taskar, B. (2002). Learning probabilistic models of link structure. *Journal of Machine Learning Research*, *3*, 679–707.

Kapoor, A., Qi, Y., Ahn, H., & Picard, R. (2005). Hyperparameter and kernel learning for graph-based semi-supervised classification. *Neural Information Processing Systems 18*.

Krishnapuram, B., Williams, D., Xue, Y., Carin, L., Hartemink, A., & Figueiredo, M. (2004). On semi-supervised classification. *Neural Information Processing Systems (NIPS)*.

Lawrence, N. D., & Jordan, M. I. (2005). Semi-supervised learning via Gaussian processes. *Advances in Neural Information Processing Systems 17* (pp. 753–760).

Minka, T. P. (2001). *A family of algorithms for approximate Bayesian inference*. Ph.D. thesis, Massachusetts Institute of Technology.

Opper, M., & Winther, O. (2005). Expectation consistent approximate inference. *Journal of Machine Learning Research*, 2117–2204.

Rasmussen, C. E., & Williams, C. K. I. (2006). *Gaussian processes for machine learning*. The MIT Press.

Seeger, M. (2003). *Bayesian Gaussian process models: PAC-Bayesian generalisation error bounds and sparse approximations*. Doctoral dissertation, University of Edinburgh.

Sindhwani, V., Chu, W., & Keerthi, S. S. (2007). Semi-supervised Gaussian process classification. *The Twentieth International Joint Conferences on Artificial Intelligence*. to appear.

Sindhwani, V., Niyogi, P., & Belkin, M. (2005). Beyound the point cloud: from transductive to semi-supervised learning. *Proceedings of the 22th International Conference on Machine Learning* (pp. 825–832).

Taskar, B., Abbeel, P., & Koller, D. (2002). Discriminative probabilistic models for relational data. *Proceedings of Conference on Uncertainty in Artificial Intelligence*.

Wagstaff, K., Cardie, C., Rogers, S., & Schroedl, S. (2001). Constrained k-means clustering with background knowledge. *Proceedings of International Conference on Machine Learning* (pp. 577–584).

Wainwright, M. J., Jaakkola, T., & Willsky, A. S. (2005). A new class of upper bounds on the log partition function. *IEEE Trans. on Information Theory*, *51*, 2313–2335.

Yu, K., Chu, W., Yu, S., Tresp, V., & Xu, Z. (2006). Stochastic relational models for discriminative link prediction. *Advances in Neural Information Processing Systems*. to appear.

Zhou, D., Bousquet, O., Lal, T., Weston, J., & Schölkopf, B. (2004). Learning with local and global consistency. *Advances in Neural Information Processing Systems 18* (pp. 321–328).

Zhu, X., Ghahramani, Z., & Lafferty, J. (2003). Semi-supervised learning using Gaussian fields and harmonic functions. *Proceedings of the 20th International Conference on Machine Learning*.
